# Assembly Fuzzy Representation on Hypergraph for Open-Set 3D Object Retrieval

**Yang Xu**[1], **Yifan Feng**[1], **Jun Zhang**[2], **Jun-Hai Yong**[1], **and Yue Gao**[1*]

[1]BNRist, THUIBCS, KLISS, BLBCI, School of Software, Tsinghua University, China
[2]Tencent AI Lab
{xuyang9610,evanfeng97}@gmail.com, junejzhang@tencent.com,
{yongjh,gaoyue}@tsinghua.edu.cn

## Abstract

The lack of object-level labels presents a significant challenge for 3D object retrieval in the open-set environment. However, part-level shapes of objects often share commonalities across categories but remain underexploited in existing retrieval methods. In this paper, we introduce the Hypergraph-Based Assembly Fuzzy Representation (HAFR) framework, which navigates the intricacies of open-set 3D object retrieval through a bottom-up lens of ***Part Assembly***. To tackle the challenge of assembly isomorphism and unification, we propose the Hypergraph Isomorphism Convolution (HIConv) for smoothing and adopt the Isomorphic Assembly Embedding (IAE) module to generate assembly embeddings with geometric-semantic consistency. To address the challenge of open-set category generalization, our method employs high-order correlations and fuzzy representation to mitigate distribution skew through the Structure Fuzzy Reconstruction (SFR) module, by constructing a leveraged hypergraph based on local certainty and global uncertainty correlations. We construct three open-set retrieval datasets for 3D objects with part-level annotations: OP-SHNP, OP-INTRA, and OP-COSEG. Extensive experiments and ablation studies on these three benchmarks show our method outperforms current state-of-the-art methods.

## 1 Introduction

With the growing accessibility of 3D data, 3D object retrieval (3DOR) has emerged as an important area of interest in computer vision [19, 12]. The key objective of 3DOR is to establish connections between query and target samples via model training. Despite significant progress in recent years enhancing the development of 3DOR, most existing methods are still based on the closed-set assumption, where all categories encountered in the testing phase have been seen during training [6]. However, training sets can not cover all potential categories in real-world applications [32], which hinders accurate retrieval for unseen categories. Although object-level categories are difficult to cover, the part-level shapes of objects across object-level categories often share commonalities [17] among objects, which may provide sufficient semantic information of the object [29, 33]. However, this part-level method still remains underexploited on assembly-based representation for open-set retrieval.

3D parts, as essential components in shape representation and analysis, have the capability to represent potential information for both semantic [30] and geometric [25] level tasks. Recently, there has been an increasing application of assembly-based methods in the realm of 3D vision [24, 37]. From a geometric perspective, current methods for 3D part assembly impose strict constraints on the

---

[*]Corresponding author

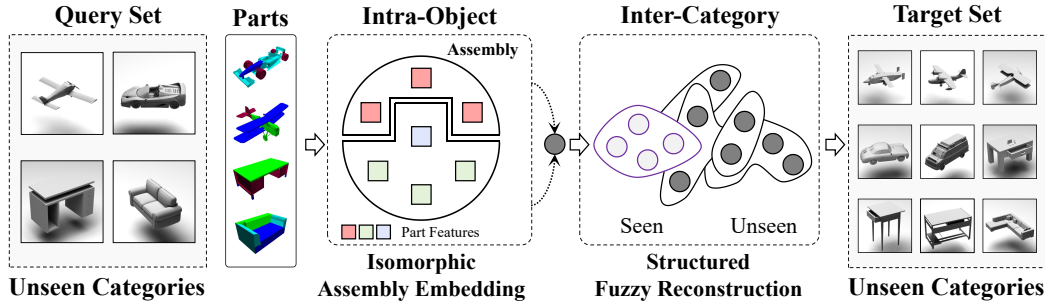

Figure 1: Illustration of the assembly-based open-set 3DOR task and proposed HAFR framework. Given 3D objects of unseen categories, our method takes several part features and generates the assembly embedding isomorphically for each object. Then fuzzy embeddings are generated via leverage propagation and fuzzy reconstruction for open-set retrieval with unseen categories generalization.

categories or quantities of parts [45], presenting a substantial challenge for implementation in an open-set environment. These methods often overlook the isomorphism and correlations between parts of the same object [20] during semantic embedding. This oversight diminishes the generalization capabilities of the representation model and exacerbates the distribution skew of unseen categories during open-set learning [47, 27]. In this paper, we explore the part-assembly representation method from both semantic and geometric perspectives to mitigate the distribution skew of unseen categories, aiming to enhance the generalization performance for open-set 3D object retrieval.

Distinct from existing 3D shape assembly or open-set learning methods, the assembly-based open-set retrieval task emphasizes global-level semantics in object representation and requires enhanced generalization of geometric assembly. This leads to several challenges for assembly-based open-set retrieval, including: **First, the difficulty of achieving assembly isomorphism for part features**. While part features can effectively capture the categorical information of objects, direct fusion for multiple parts may lead to inconsistencies arising from geometric factors, including the order of input and the presence of repeated parts. Consequently, there is a strong motivation to achieve assembly isomorphism for part features with geometric-semantic consistency. **Second, the difficulty in achieving assembly unification across different parts** entails mapping and integrating part embeddings from the local-part space to the global-object space. **Third, the difficulty in open-set category generalization against distribution skew**, which requires spatial propagation and generalization for object embeddings from the seen certainty to unseen uncertainty space.

Addressing the aforementioned challenges, we explore a method for open-set retrieval tasks through a bottom-up lens of ***Part Assembly***. As shown in Figure 1, we introduce the Hypergraph-based Assembly Fuzzy Representation (HAFR) framework for assembly-based open-set 3DOR. On one hand, to tackle the challenge of assembly isomorphism and unification, we first propose the Hypergraph Isomorphism Convolution (HIConv) layer for feature smoothing, and then we adopt the Isomorphic Assembly Embedding (IAE) module for embedding integration with geometric-semantic consistency. On the other hand, to overcome the difficulty in category generalization, we construct a leverage hypergraph based on the local-certainty and global-uncertainty correlations. This structure captures potential leveraged correlations between seen and unseen categories for propagation. Besides, we adopt the Structure Fuzzy Reconstruction (SFR) module to exploit the fuzzy representation approach for open-set category generalization. Our contributions are summarized as follows:

- We explore a method to navigate the intricacies of open-set 3D object retrieval through a bottom-up lens of *Part Assembly*, and we construct three 3D point cloud datasets with multiple part annotations for benchmarking.

- We propose the HAFR framework for assembly-based open-set 3D object retrieval tasks, including the Isomorphic Assembly Embedding (IAE) and the Structured Fuzzy Reconstruction (SFR) modules, which are designed to generate assembly embeddings with geometric-semantic consistency and overcome the distribution skew of unseen categories.

- We propose the Hypergraph Isomorphism Convolution (HIConv) and a leverage hypergraph structure to capture the high-order correlations within and among objects, utilizing them for assembly isomorphism and open-set category generalization.

- Extensive experiments are conducted on the three benchmarks for evaluation, demonstrating the superiority of HAFR over current state-of-the-art 3D object retrieval methods.

## 2  Related Work

### 2.1  3D Object Retrieval

Most current 3D object retrieval methods operate under the closed-set assumption, meaning that the training and testing sets exhibit the same distribution of categories. These methods are divided into single-modal and multi-modal types. Single-modal 3D object retrieval focuses on identifying similar objects within a single modality of 3D data. For example, [31] and [35] use a view-based graph model to generate aggregated embeddings from multi-view data. [13] introduces a triplet-center loss to cluster objects of the same category and separate those of different categories. HGNN [9] employs a hypergraph-based structure to capture high-order correlations among objects for improved embeddings. Multi-modal retrieval methods [26, 22, 43, 44, 7, 2] use weighted fusion or feature fusion networks to aggregate embeddings from different modality-specific features. Additionally, CMCL [15] proposes a cross-modal center loss to minimize differences across various 3D modalities using common center embeddings.

### 2.2  Open-Set Learning

Open-set learning focuses on conducting machine learning research in scenarios where key factors are variable [47]. [32] introduces the Semantic Shift Benchmark (SSB) for open-set recognition. [46] proposes a "none-of-above" classifier to determine if a sample belongs to known categories. [5] presents an adversarial method to minimize the overlap between known and unknown distributions. Additionally, several open-set recognition methods for 3D object learning have been proposed [3, 16, 1, 48]. However, retrieval tasks in open-set scenarios are more practical than recognition due to the fundamentals of representation. Only a few methods [8, 23, 40, 39, 38] tackle the open-set 3DOR task, focusing on structure learning networks while neglecting the intricacies of the open-set environment.

### 2.3  3D Shape Assembly

Much research in computer graphics has concentrated on assembly representation for the reconstruction [37], analysis [34], and generation [17] of 3D shapes. As for the assembly-related method for retrieval, most existing methods primarily focus on establishing relationships between parts of one object and parts of other objects [10], then utilize these connections for shape analysis [21] or part retrieval [4]. Although these methods have achieved satisfactory results, they rarely concentrate on the higher-order semantic connections from parts to objects, which makes them challenging to apply in 3D object retrieval. Additionally, their geometric constraints also limit their application in open-set environments.

## 3  Problem Setup

### 3.1  Open-Set 3D Object Retrieval

Given 3D objects from the query set $\mathcal{D}_q$, the 3D object retrieval (3DOR) task is to find similar samples from the target set $\mathcal{D}_t$. The core approach for the 3DOR task is to learn the relationship between query samples and target samples from the training set $\mathcal{D}_{trn}$. Each 3D object can be denoted as $(s_i, y_i)$, the $y_i \in \mathcal{Y} = \{c_j\}_{j=1}^Y$ is the category label associated with the 3D object sample $s_i$.

In the open-set environment for 3DOR, all categories of samples in the testing set have not been learned in the training set, each retrieval sample is from unseen categories for the model, termed as *Open-Set Retrieval*. Specifically, the open-set settings means that the testing set $\mathcal{D}_{tes} = \{\mathcal{D}_q, \mathcal{D}_t\}$ and the training set $\mathcal{D}_{trn}$ are drawn from the different distributions. For the testing set $\mathcal{D}_{tes} =$

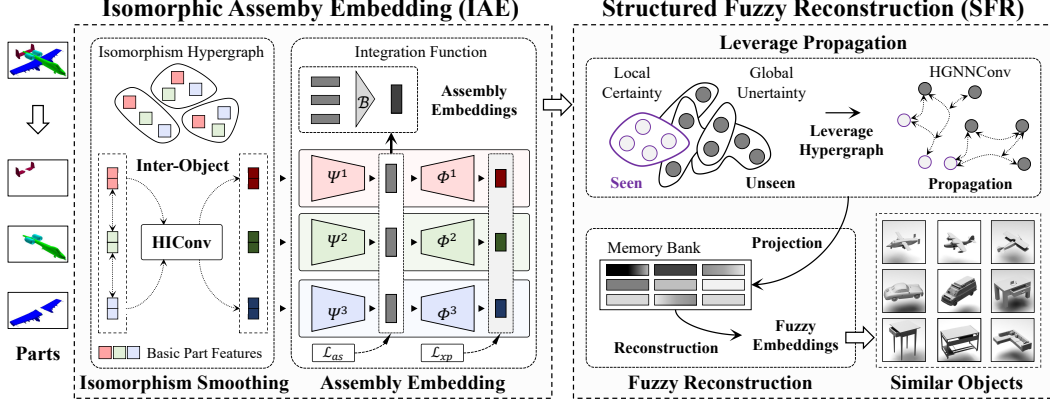

Figure 2: An overview of the Hypergraph-Based Assembly Fuzzy Representation (HAFR) framework for assembly-based open-set 3D object retrieval. Our framework is composed of the Assembled Isomorphism Embedding (IAE) and Structured Fuzzy Reconstruction (SFR) modules, which are designed for geometric-semantic consistent integration and fuzzy-aware generalization, respectively.

$\{(s_i, \hat{y}_i)\}_{i=1}^T = \{\mathcal{D}_q, \mathcal{D}_t\}$ and the training set $\mathcal{D}_{trn} = \{(s_i, y_i)\}_{i=1}^L$, the category space of them are not the same indicating $\hat{y}_i \in \hat{\mathcal{Y}} = \{\hat{c}_j\}_{j=1}^{\hat{Y}}$, $y_i \in \mathcal{Y} = \{c_j\}_{j=1}^Y$, and $\hat{\mathcal{Y}} \cap \mathcal{Y} = \varnothing$.

## 3.2 Assembly Representation for Retrieval

Typically, each 3D object can be decomposed into multiple parts based on its shape and semantic information, and it can be regarded as a model assembled from these parts, *i.e.*, the doors, roofs, hoods, and wheels of cars, the seats, backs, arms, and legs of the chairs. Face the emergence of the open-set environment with incomplete or missing class labels, the assembly representation based on multiple local parts may provide enough semantic information than global object features for retrieval. We termed this presentation for open-set 3DOR as *Assembly Representation*:

$$s_i \in \mathcal{S} = \{\{p_i^r\}_{r=1}^P\}_{i=1}^N, \tag{1}$$

where $\mathcal{S}$ denote the set of 3D objects, $s_i = \{p_i^r\}_{r=1}^P$ denotes a 3D object represented by $P$ semantic parts, $N$ denotes the number of samples in $\mathcal{S}$.

Consequently, the assembly-based open-set 3DOR aims to design a method to retrieve similar samples of query in the testing set $\mathcal{D}_{tes} = \{(\{p_i^r\}_{r=1}^P, \hat{y}_i)\}_{i=1}^T = \{\mathcal{D}_q, \mathcal{D}_t\}$, based on the data and knowledge in the training set $\mathcal{D}_{trn} = \{(\{p_i^r\}_{r=1}^P, y_i)\}_{i=1}^L$. The assembly-based open-set 3DOR by multiple parts aims to minimize the expected risk:

$$f^* = \underset{f \in \mathcal{H}}{arg\,min}\, \mathbb{E}_{(D_i, D_j) \sim (\mathcal{D}_q, \mathcal{D}_t)} \left[ \chi_{\{\hat{y}_i \neq \hat{y}_j\}} e^{-\|f(\{p_i^r\}_{r=1}^P) - f(\{p_j^r\}_{r=1}^P)\|_2} \right.$$
$$\left. + \chi_{\{\hat{y}_i = \hat{y}_j\}} (1 - e^{-\|f(\{p_i^r\}_{r=1}^P) - f(\{p_j^r\}_{r=1}^P)\|_2}) \right], \tag{2}$$

where $D_i = (\{p_i^r\}_{r=1}^P, n_i, \hat{y}_i)$ and $D_j = (\{p_j^r\}_{r=1}^P, n_j, \hat{y}_j)$ are the object samples selected from the query set $\mathcal{D}_q$ and target set $\mathcal{D}_t$. $\chi_{\{\cdot\}}$ denotes the indicator function, which evaluates to 1 when the specified condition holds true and 0 otherwise. The embedding function $f := \{p_i^r\}_{r=1}^P \to v_i$ maps multiple parts $\{p_j^r\}_{r=1}^P$ of an object to an assembly representation vector $v_i \in \mathbb{R}^d$, facilitating similarity-based retrieval. $\mathcal{H}$ denotes the hypothesis space of the embedding function. The $\mathcal{L}_2$ norm function $\|\cdot\|_2$ is used as a distance metric to measure the Euclidean distance between two vectors.

## 4 Methodology

### 4.1 Overall Framework

As shown in Figure 2, the proposed Hypergraph-Based Assembly Fuzzy Representation (HAFR) framework is composed of two modules: *Isomorphic Assembly Embedding (IAE)* and *Structured*

*Fuzzy Reconstruction (SFR)*. The framework takes the basic features of different parts as input. In the IAE stage, the multiple features are assembled isomorphically by the Hypergraph Isomorphism Convolution (HIConv), and the assembly embeddings are generated from multiple parts. Next, in the SFR stage, the leverage hypergraph structure is constructed based on the local-certainty and global-uncertainty correlations. Guided by this structured open-set distribution, hypergraph convolution is adopted for propagation implicitly from seen categories to unseen categories. Finally, the memory bank is adopted for fuzzy-aware reconstruction and fuzzy embedding generation to further overcome openness during open-set retrieval.

## 4.2 Isomorphic Assembly Embedding

The IAE module is designed here to obtain assembly embeddings with geometric-semantic consistency from multiple parts. Specifically, the IAE comprises a hypergraph-based iso-morphism layer and assembly auto-encoders as shown in the left side of Figure 2. We proposed the *Hypergraph Isomorphism Convolution (HIConv)* in the layer to impart geometric-semantic consistency to part features, overcoming the bias introduced by the order of parts during assembly representation. The assembly auto-encoders are utilized to get the assembly embedding from multiple part features.

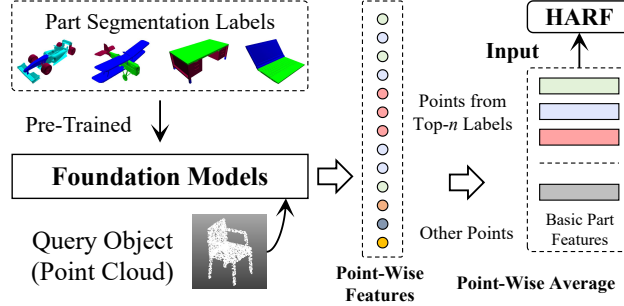

Figure 3: Illustration of input basic part features for HAFR.

The IAE module takes the basic part features $\{\{f_i^r\}_{r=1}^P\}_{i=1}^N$ ($f_i^r \in \mathbb{R}^{N \times d_f}$) of $N$ objects with $P$ parts. In this paper, we use the average point feature of each region extracted by 3D point cloud part segmentation network [28] as shown in Figure 4.1. The hypergraph isomorphism layer constructs an assembly hypergraph structure $\mathcal{G}_a$ and generates isomorphism embeddings $c_i^r$ under the guidance of this structure. The assembly auto-encoders $\mathcal{A}_a$ encode the multi-part features of 3D objects and get the assembly embeddings $u_i$ by integration function.

### 4.2.1 Isomorphism Smoothing

Although part features have the potential to adequately represent the category information of every object, direct fusion for multiple parts may lead to inconsistencies arising from geometric factors, including the order of input and the presence of repeated parts. The HIConv are utilized to impart geometric-semantic consistency during part assembly, aiming to generate isomorphism embeddings from independent-part to correlated-object distribution, by the correlation-based smoothing under the guidance of the assembly hypergraph structure.

A hypergraph can be represented as $\mathcal{G} = \{\mathcal{V}, \mathcal{E}\}$, where $\mathcal{V}$ and $\mathcal{E}$ are the vertex set and the hyperedge set, respectively. In the hypergraph of HIConv, the basic part features $f_i^p$ are treated as the vertices $\mathbf{X}_f = \bigcup_{d=1}^D \{f_i^d\}_{i=1}^N$. Then the isomorphism hyperedges $\mathcal{E}_o$ are constructed as the subset of vertices that are from the same object:

$$\mathcal{E}_o = \{\mathcal{O}_v(i) \mid i \in \{1, \cdots, N\}\} \tag{3}$$

where $\mathcal{O}_v(i)$ denotes all vertices of the $i$-th object. In this way, we obtain $N$ hyperedges and $N$ is the number of objects. The assembly hypergraph is constructed as $\mathcal{G}_a = \{\mathbf{X}_f, \mathcal{E}_o\}$ after getting vertices and hyperedges.

After the construction of the assembly hypergraph, the HIConv is designed to bridge the geometry and semantic correlations for isomorphic smoothing. For the convenience of convolution, we use the incidence matrix $\mathbf{H} \in \{0, 1\}^{|\mathcal{V}| \times |\mathcal{E}|}$ to represent the hypergraph, where the hyperedges are the columns of $\mathbf{H}$, and $\mathbf{H}(v, e) = 1$ if vertex $v$ are contained in hyperedge $e$. Inspired by [18] and [11], the HIConv is designed to leverage the geometric-semantic collaborative information under the

guidance of assembly hypergraph:

$$\tilde{\mathbf{X}}_f = \text{MLP}\left((1+\epsilon)\mathbf{X}_f + \sigma(\mathbf{D}_v^{-\frac{1}{2}}\mathbf{H}\mathbf{D}_e^{-1}\mathbf{H}^\top \mathbf{D}_v^{-\frac{1}{2}}\mathbf{X}_f \Theta_{HIConv})\right), \quad (4)$$

where $\mathbf{H}$ denotes the incidence matrix of the hypergraph $\mathcal{G}_a$, respectively. $\mathbf{D}_v$ and $\mathbf{D}_e$ are the diagonal degree matrices for vertices and hyperedges, respectively. $\epsilon$ denotes a learnable parameter and $\Theta_{HIConv}$ is the learnable matrix for HIConv. After the HIConv, the isomorphism embeddings $\tilde{\mathbf{X}}_f = \{\{c_i^r\}_{r=1}^P\}_{i=1}^N$ are generated from independent-part to correlated-object space.

### 4.2.2 Assembly Embedding

The IAE module first utilizes assembly auto-encoders $\mathcal{A}_a$ to encode the isomorphism embeddings into a latent code space for each part, then employs the isomorphism loss to pull the different part codes together to ensure encoded embeddings from the same object together. Besides, the intra-part and inter-part reconstruction loss are proposed to reduce information loss during compression.

Specifically for the $\mathcal{A}_a$, we have $u_i^r = \Psi^r(c_i^r)$ and $\hat{c}_i^r = \Phi^r(u_i^r)$, where $c_i^r \in \mathbb{R}^{d_f}$ denote isomorphism embeddings of parts, $u_i^r \in \mathbb{R}^{d_u}$ denote the unified embeddings compressed from isomorphism embeddings of different parts. The encoder and decoder are defined as $\Psi^r := \mathbb{S}_p \to \mathbb{S}_a$ and $\Phi^r := \mathbb{S}_a \to \mathbb{S}_p$, which map the representation between local-part space $\mathbb{S}^p$ to global-assembly space $\mathbb{S}^a$.

After the generation of unified embeddings $u_i^r$ for each part, the IAE module employs an integration function $\mathcal{B}(\cdot)$ to obtain the assembly embeddings integrated from all parts of the same object:

$$u_i = \mathcal{B}(\{u_i^r\}_{r=1}^P) \quad (5)$$

### 4.2.3 Loss Function for the IAE

For better isomorphism smoothing and assembly embedding, we adopt the Assembly Loss $\mathcal{L}_{as}$ and the Cross-Part Loss $\mathcal{L}_{xp}$ for the IAE module, which is designed to pull the distance and prompt the generalization performance across parts, respectively:

$$\mathcal{L}_{as} = \frac{2}{R(R-1)}\sum_{k=1}^R \sum_{l=k+1}^R \|u_i^k - u_i^l\|_2, \quad (6)$$

$$\mathcal{L}_{xp} = \frac{2}{R(R-1)}\sum_{k=1, l\neq k}^D \left(\|c_i^k - \hat{c}_i^k\|_2 + \|c_i^k - \Phi^l\left(\Psi^k\left(c_i^l\right)\right)\|_2\right), \quad (7)$$

where $\|\cdot\|_2$ is the $\mathcal{L}_2$ norm, $u_i^k$ and $u_i^l$ are both the unified embeddings but from different parts of the same object, $\Psi^k$ is the encoder of $k$-th part and $\Phi^l$ is the decoder of $l$-th part.

The loss function for IAE is constructed by the balanced combination of the two losses: $\mathcal{L}_{IAE} = \alpha\mathcal{L}_{as} + (1-\alpha)\mathcal{L}_{xp}$, where $\alpha$ is the hyper-parameter to trade-off between them.

## 4.3 Structured Fuzzy Reconstruction

Although the IAE module generated the assembly embeddings from multiple parts, the distribution skew across seen and unseen categories still affects the performance of open-set retrieval. As shown in Figure 2, we proposed the SFR module for generalization. The SFR module first constructs a leverage hypergraph to model the local-certainty and global-uncertainty correlations. Then SFR employs hypergraph convolution for propagation from seen categories to unseen categories based on the implicit leveraged structure. After that, the memory bank is adopted to reconstruct the propagation embeddings into fuzzy space to further overcome openness during open-set retrieval.

### 4.3.1 Leverage Propagation

To get the most out of potential correlations from seen categories, the leverage hypergraph is designed here. As shown in Figure 2, the assembly embeddings $u_i$ are treated as the vertices $\mathbf{X}_u = \{u_i\}_{i=1}^N$ in the leverage hypergraph, and the hyperedges are constructed from two perspectives: local-certainty and global-uncertainty correlations.

The local-certainty hyperedges $\mathcal{E}_c$ are constructed based on the category observability, which is defined as $\mathcal{E}_c = \{C_v(y) \mid y \in \mathcal{Y}\}$, where $C_v(y)$ denotes the subset of vertices that belong to the

seen categories $\mathcal{Y}$. For the global-uncertainty hyperedges $\mathcal{E}_u$, we construct them by linking each vertex and its $K - 1$ neighbor vertices: $\mathcal{E}_u = \{K_{\text{KNN}_k}(v) \mid v \in \mathcal{V}\}$, where $K_{\text{KNN}_k}(v)$ denotes the top-$k$ nearest neighbor set of vertex $v$. In this way, we obtain one local-certainty hyperedge and $N$ global-uncertainty hyperedge. The leverage hypergraph is constructed by $\mathcal{G}_{lev} = \{\mathbf{X}_u, \mathcal{E}_c \cup \mathcal{E}_u\}$.

After the construction of the leverage hypergraph, we utilize the modified hypergraph convolution from [11] for embedding propagation.

$$\tilde{\mathbf{X}}_u = \sigma\left(\mathbf{D}_v^{-\frac{1}{2}}\mathbf{H}\mathbf{D}_e^{-1}\mathbf{H}^{\top}\mathbf{D}_v^{-\frac{1}{2}}\mathbf{X}_u\mathbf{\Theta}_{lev}\right),\qquad(8)$$

where $\mathbf{H}$ denotes the incidence matrix of the leverage hypergraph $\mathcal{G}_{lev}$. $\mathbf{D}_v$ and $\mathbf{D}_e$ are the diagonal degree matrices for vertices and hyperedges, respectively. $\mathbf{\Theta}_{lev}$ denotes the learnable matrix for the HGNNConv, $\tilde{\mathbf{X}}_u$ are the propagation embeddings and $\tilde{\mathbf{X}}_u = \{p_i\}_{i=1}^N$.

### 4.3.2 Fuzzy Reconstruction

To overcome the distribution skew caused by the open-set environment, the memory bank is adopted here to reconstruct the propagation embeddings to fuzzy space. The memory bank is designed to store a large amount of fuzzy representations with uniform distribution. Specifically, the memory bank $\mathcal{M}$ is composed of $Z$ invariant memory anchors $m_j$ for 3D objects.

$$\mathcal{M} = \{m_j \in \mathbb{R}^u \mid j = 1, \cdots, Z\}\qquad(9)$$

Given the propagation embedding $\tilde{u}_i$ of the each 3D object, the activation score $s_{i,j}$ are calculated for every memory anchor $m_j$ in $\mathcal{M}$ by $s_{i,j} = \|\tilde{u}_i - m_j\|_2$, where $\|\cdot\|_2$ is the $\mathcal{L}_2$ norm for distance metric, $s_{i,j}$ denotes the activation score of each anchor. Then we use the normalization of activation scores $s'_{i,j}$ to rebuild the propagation embedding into fuzzy space and get fuzzy embeddings $z_i$ by:

$$z_i = \sum\nolimits_{j=1}^Z s'_{i,j}m_j\qquad(10)$$

### 4.3.3 Loss Function for the SFR

In the SFR stage, we adopt the Cross-Entropy Loss $\mathcal{L}_{ce}$ and Fuzzy Reconstruction Loss $\mathcal{L}_{fz}$:

$$\mathcal{L}_{fz} = \left\|\tilde{u}_i - z_i\right\|_2,\qquad(11)$$

$$\mathcal{L}_{ce} = -\sum\nolimits_{k=1}^Y \left(n_{i,k}\log(p_{i,k}) + n_{i,k}\log(q_{i,k})\right),\qquad(12)$$

where $p_{i,k} = \frac{e^{\tilde{v}_{i,k}}}{\sum_{k=1}^Y e^{\tilde{u}_{i,k}}}$ and $q_{i,k} = \frac{e^{z_{i,k}}}{\sum_{k=1}^Y e^{z_{i,k}}}$ denote the prediction scores of fuzzy embeddings that the image belongs to the $k$-th category. $n_{i,k}$ is the $k$-th value of one-hot encoded seen category labels, $Y$ is the number of seen categories.

The loss function for SFR is constructed by the balanced combination of the two losses: $\mathcal{L}_{SFR} = \beta\mathcal{L}_{fz} + (1 - \beta)\mathcal{L}_{ce}$, where $\beta$ is the hyper-parameter to trade-off between them.

## 5 Experiments

### 5.1 Experimental Settings

**OpenPart Datasets.** We construct three datasets for assembly-based open-set 3D object retrieval (OpenPart datasets), including OP-SHNP, OP-INTRA, OP-COSEG based on the public dataset ShapeNetPart [42], IntrA [41], and COSEG [34]. We sampled the point cloud from the triangular surface for each dataset. As shown in Table 1, the classes of these datasets are split into seen and unseen classes for training and testing, respectively. Each class contains three to five parts. Specifically, the detailed descriptions of the datasets and parts segmentation are shown in Appendix B.

Table 1: The statistics of the three OpenPart datasets.

| Dataset | | OP-SHNP | OP-INTRA | OP-COSEG |
|---|---|---|---|---|
| Average Parts/Sample | | 3.2 | 3.1 | 2.3 |
| Category | All | 16 | 3 | 9 |
| | Seen | 6 | 1 | 3 |
| | Unseen | 10 | 2 | 6 |
| Number | Train | 5804 | 116 | 240 |
| | Retrieval | 6142 | 2318 | 802 |
| | Query | 945 | 302 | 85 |
| | Target | 5197 | 2016 | 717 |

Table 2: Quantitative results of retrieval on the OP-SHNP, OP-INTRA, and OP-COSEG datasets.

| Method | OP-SHNP | | | OP-INTRA | | | OP-COSEG | | |
|---|---|---|---|---|---|---|---|---|---|
| | mAP↑ | NDCG↑ | ANMRR↓ | mAP↑ | NDCG↑ | ANMRR↓ | mAP↑ | NDCG↑ | ANMRR↓ |
| MMJN | 0.5685 | 0.5856 | 0.2599 | 0.5465 | 0.5898 | 0.5053 | 0.6394 | 0.7623 | 0.4314 |
| TCL | 0.5683 | 0.5861 | 0.2608 | 0.5467 | 0.5970 | 0.5059 | 0.6285 | 0.7543 | 0.4401 |
| SDML | 0.5699 | 0.5870 | 0.2593 | 0.5456 | 0.5944 | 0.5064 | 0.6328 | 0.7638 | 0.4422 |
| MMSAE | 0.5637 | 0.5824 | 0.2659 | 0.5452 | 0.5919 | 0.5056 | 0.6350 | 0.7555 | 0.4334 |
| PROSER | 0.5687 | 0.5861 | 0.2607 | 0.5462 | 0.5943 | 0.5059 | 0.6343 | 0.7605 | 0.4348 |
| HGM$^2$R | 0.5736 | 0.5886 | 0.2549 | 0.5545 | 0.6019 | 0.4928 | 0.6452 | 0.7627 | 0.4355 |
| **Ours** | **0.5947** | **0.5916** | **0.2239** | **0.5750** | **0.6382** | **0.4797** | **0.7015** | **0.7629** | **0.3604** |

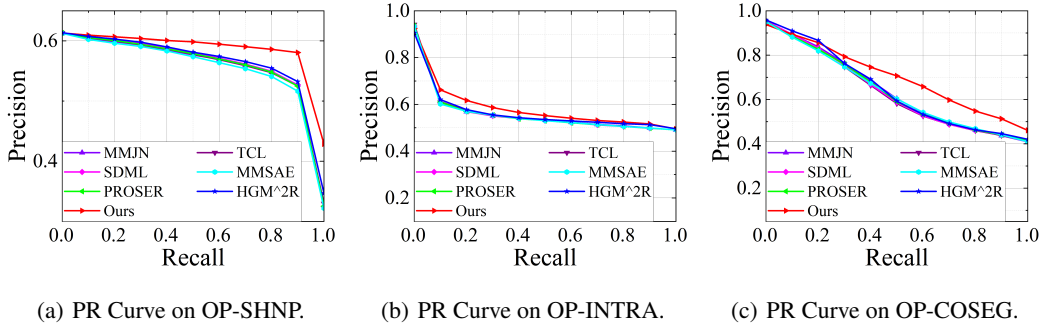

(a) PR Curve on OP-SHNP.    (b) PR Curve on OP-INTRA.    (c) PR Curve on OP-COSEG.

Figure 4: The Precision-Recall Curves for comparison on the three datasets, respectively.

**Implemental Details.** The random seeds are fixed to 2024 in this paper for a fair comparison. The basic features are extracted by PointNet [28] and each part feature is obtained by the average point feature of each region, we use the top four parts for each object in this paper. We set $\alpha = 0.5$ in $\mathcal{L}_{IAE}$ and $\beta = 0.9$ in $\mathcal{L}_{SFR}$. The IAE is trained for 40 epochs on learning rate $lr = 0.1$, and the SFR is trained for 30 epochs on $lr = 0.001$. The detailed implements of the HAFR framework are provided in Appendix C.

## 5.2 Retrieval Performance

**Compared Methods.** Since no methods are specifically designed for assembly-based open-set 3DOR, we refine the current state-of-the-art close-set 3DOR methods (MMJN [26], TCL [13], SDML [14], MMSAE [36]), and open-set 3D learning methods (PROSER [46], HGM$^2$R [8]), then we refine the multi-modal fusion module with multi-part fusion for each method. We provide detailed implements of compared methods in Appendix D

**Evaluation Metrics.** For a fair comparison, we use standard retrieval metrics, including Mean Average Precision (mAP), Normalized Discounted Cumulative Gain (NDCG), Average Normalized Modified Retrieval Rank (ANMRR), and the Precision-Recall Curve (PR-Curve). For the mAP and NDCG metrics, higher scores are better. For the ANMRR metric, a lower score is better.

**Comparsion Analysis.** As shown in Table 2, we evaluate the assembly-based open-set retrieval results from HAFR framework and other state-of-the-art compared methods. Quantitative results demonstrate the superiority of our method over the other methods on all three datasets. In particular, on the OP-COSEG dataset, our method achieves $0.7015$ mAP with about $8.7\%$ improvements compared with the second-best method (HGM$^2$R). We also compare the Precision-Recall Curves (PR-Curve) which evaluate the trade-off between precision and recall at different thresholds. A larger area between the curve and the axes indicates better performance for retrieval, our method outperforms all other retrieval methods as shown in Figure 4.

The superior performance in the comparison indicates that by the proposed IAE and SFR modules, the proposed HAFR framework can better utilize the potential semantic information among part

Table 3: Ablation studies of retrieval on the OP-SHNP, OP-INTRA, and OP-COSEG datasets.

| Ablation | OP-SHNP | | | OP-INTRA | | | OP-COSEG | | |
|---|---|---|---|---|---|---|---|---|---|
| | mAP↑ | NDCG↑ | ANMRR↓ | mAP↑ | NDCG↑ | ANMRR↓ | mAP↑ | NDCG↑ | ANMRR↓ |
| HIConv→GIN | 0.5761 | 0.5826 | 0.2465 | 0.5567 | 0.6050 | 0.4947 | 0.6782 | 0.7625 | 0.4001 |
| HIConv→MLP | 0.5837 | 0.5854 | 0.2357 | 0.5523 | 0.6035 | 0.4948 | 0.6505 | 0.7664 | 0.4334 |
| IAE w/o $\mathcal{L}_{as}$ | 0.5767 | 0.5838 | 0.2464 | 0.5560 | 0.6077 | 0.4946 | 0.6829 | 0.7542 | 0.3823 |
| IAE w/o $\mathcal{L}_{xp}$ | 0.5767 | 0.5840 | 0.2464 | 0.5561 | 0.6078 | 0.4944 | 0.6828 | 0.7542 | 0.3824 |
| SFR w/o $\mathcal{G}_{lev}$ | 0.5755 | 0.5825 | 0.2474 | 0.5545 | 0.6019 | 0.4928 | 0.6887 | 0.7540 | 0.3768 |
| MLP-based SFR | 0.5636 | 0.5851 | 0.2665 | 0.5429 | 0.5814 | 0.4997 | 0.6662 | 0.7610 | 0.4062 |
| GCN-based SFR | 0.5679 | 0.5845 | 0.2603 | 0.5430 | 0.5854 | 0.4982 | 0.6674 | 0.7618 | 0.4072 |
| **IAE+SFR** | **0.5947** | **0.5916** | **0.2239** | **0.5750** | **0.6382** | **0.4797** | **0.7015** | **0.7629** | **0.3604** |

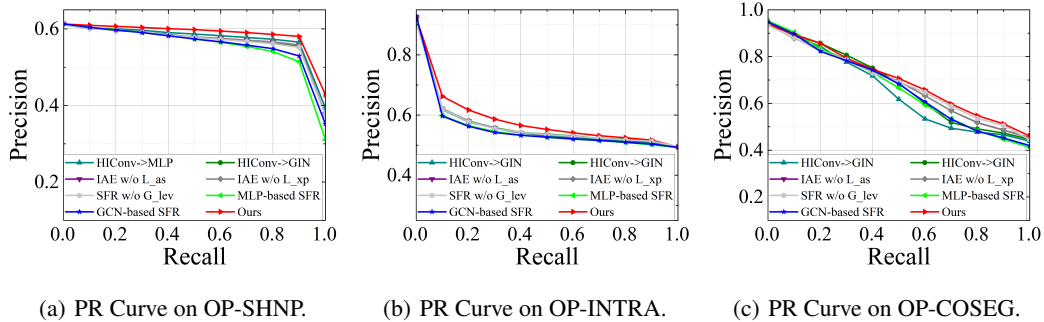

(a) PR Curve on OP-SHNP.  (b) PR Curve on OP-INTRA.  (c) PR Curve on OP-COSEG.

Figure 5: The Precision-Recall Curves of the ablation studies on the three datasets, respectively.

features to fully represent 3D objects and generalize them to unseen categories. Notably, for the OP-INTRA dataset with only one category and the least parts in the training set, our method still achieves performance improvements in the retrieval of unseen categories as shown in Table 2 and Figure 4. This indicates that our method has minimal dependence on the performance of classifiers during basic feature extraction. In open-set environments, where training on all potential categories is infeasible, our HAFR method through *Part Assembly* demonstrates superior open-set adaptability. We provide more visualizations and results in Appendix D.

## 5.3 Ablation Studies

We conduct ablation studies to verify the effectiveness of modules in the proposed framework. For the IAE stage, we first remove the Hypergraph Isomorphism Convolution (HIConv) layer for comparison. Specifically, we replace it with the Graph Isomorphism Convolution (HIConv→GIN) and MLP (HIConv→MLP). Then we compared it with the method without the Assembly Loss (IAE w/o $\mathcal{L}_{as}$) or the Cross-Part Loss(IAE w/o $\mathcal{L}_{xp}$). As shown in Table 3 and Figure 5, replacing HIConv or removing parts of the $\mathcal{L}_{IAE}$ loss significantly degrades the performance of IAE. For the HIConv, this indicates that naive part assembly is dependent on geometric or semantic factors such as order and quantity. Removing isomorphism smoothing results in geometric or semantic inconsistencies, thereby degrading the retrieval performance by the worse assembly embeddings. As for the SFR module, we compare the hypergraph structure without the leveraged structure (SFR w/o $\mathcal{G}_{lev}$) and we also replace the hypergraph-based structure learning with MLP (MLP-based SFR) and GCN (GCN-based SFR). Besides, we remove the memory bank to verify the effectiveness of fuzzy reconstruction.

Table 3 and Figure 5 show that the proposed full SFR module outperforms all the other ablative structures, these results show that the proposed leverage propagation and fuzzy reconstruction approach can effectively utilize the high-order correlations between seen and unseen categories for open-set generalization. Besides, we can observe that the complete combination of IAE and SFR yields the best performance. These quantitative and qualitative results indicate that the proposed HAFR framework effectively achieves part-level assembly isomorphism and unification while mitigating distribution skew from seen certainty to unseen uncertainty at the object level.

# 6 Conclusion

In this paper, we introduce the Hypergraph-Based Assembly Fuzzy Representation (HAFR) framework, which navigates the intricacies of open-set 3D object retrieval through a bottom-up lens of *Part Assembly*. Specifically, we propose the Hypergraph Isomorphism Convolusion (HIConv) and adopt the Isomorphic Assembly Embedding (IAE) module for assembly isomorphism and unification, generating the integration embeddings with geometric-semantic consistency. Besides, we employ the Structure Fuzzy Reconstruction (SFR) approach to exploit high-order correlations among objects and fuzzify representations for open-set category generalization. This module constructs a leveraged hypergraph based on local-certainty and global-uncertainty correlations to mitigate distribution skew. We construct three open-set retrieval datasets for 3D objects with part-level annotations, *i,e.*, OP-SHNP, OP-INTRA, and OP-COSEG. Extensive experiments and ablation studies on these three benchmarks show our method outperforms current state-of-the-art methods. However, due to data limitations, this paper does not currently consider the assembly fuzzy representation for varying numbers of parts, which will be a focus of our future research. We believe this paper provides a novel perspective for open-set retrieval by exploring from local to global levels.

# 7 Acknowledgement

This work was supported by Beijing Natural Science Foundation (No. L242167), CCF-Tencent Open Research Fund, and Jiangxi Provincial Natural Science Foundation (20224ACB218002).

# References

[1] Antonio Alliegro, Francesco Cappio Borlino, and Tatiana Tommasi. Towards open set 3d learning: A benchmark on object point clouds. *arXiv preprint arXiv:2207.11554*, 2022.

[2] Song Bai, Feihu Zhang, and Philip HS Torr. Hypergraph convolution and hypergraph attention. *Pattern Recognition*, 110:107637, 2021.

[3] Abhijit Bendale and Terrance E Boult. Towards open set deep networks. In *IEEE/CVF Conference on Computer Vision and Pattern Recognition*, pages 1563–1572, 2016.

[4] Siddhartha Chaudhuri, Evangelos Kalogerakis, Leonidas Guibas, and Vladlen Koltun. Probabilistic reasoning for assembly-based 3d modeling. In *ACM SIGGRAPH*, pages 1–10. ACM New York, NY, USA, 2011.

[5] Guangyao Chen, Peixi Peng, Xiangqian Wang, and Yonghong Tian. Adversarial reciprocal points learning for open set recognition. *IEEE Transactions on Pattern Analysis and Machine Intelligence*, 44(11):8065–8081, 2021.

[6] Wei Chen, Yu Liu, Weiping Wang, Erwin M Bakker, Theodoros Georgiou, Paul Fieguth, Li Liu, and Michael S Lew. Deep learning for instance retrieval: A survey. *IEEE Transactions on Pattern Analysis and Machine Intelligence*, 2022.

[7] Yihe Dong, Will Sawin, and Yoshua Bengio. Hnhn: Hypergraph networks with hyperedge neurons. *arXiv preprint arXiv:2006.12278*, 2020.

[8] Yifan Feng, Shuyi Ji, Yu-Shen Liu, Shaoyi Du, Qionghai Dai, and Yue Gao. Hypergraph-based multi-modal representation for open-set 3d object retrieval. *IEEE Transactions on Pattern Analysis and Machine Intelligence*, 46(4):2206–2223, 2023.

[9] Yifan Feng, Haoxuan You, Zizhao Zhang, Rongrong Ji, and Yue Gao. Hypergraph neural networks. In *AAAI Conference on Artificial Intelligence*, pages 3558–3565, 2019.

[10] Thomas Funkhouser, Michael Kazhdan, Philip Shilane, Patrick Min, William Kiefer, Ayellet Tal, Szymon Rusinkiewicz, and David Dobkin. Modeling by example. *ACM Transactions on Graphics*, 23(3):652–663, 2004.

[11] Yue Gao, Yifan Feng, Shuyi Ji, and Rongrong Ji. Hgnn+: General hypergraph neural networks. *IEEE Transactions on Pattern Analysis and Machine Intelligence*, 45(3):3181–3199, 2022.

[12] Yue Gao, Meng Wang, Dacheng Tao, Rongrong Ji, and Qionghai Dai. 3-d object retrieval and recognition with hypergraph analysis. *IEEE Transactions on Image Processing*, 21(9):4290–4303, 2012.

[13] Xinwei He, Yang Zhou, Zhichao Zhou, Song Bai, and Xiang Bai. Triplet-center loss for multi-view 3d object retrieval. In *IEEE/CVF Conference on Computer Vision and Pattern Recognition*, pages 1945–1954, 2018.

[14] Peng Hu, Liangli Zhen, Dezhong Peng, and Pei Liu. Scalable deep multimodal learning for cross-modal retrieval. In *Annual Conference of the Association for Computing Machinery Special Interest Group in Information Retrieval*, pages 635–644, 2019.

[15] Longlong Jing, Elahe Vahdani, Jiaxing Tan, and Yingli Tian. Cross-modal center loss for 3d cross-modal retrieval. In *IEEE/CVF Conference on Computer Vision and Pattern Recognition*, pages 3142–3151, 2021.

[16] KJ Joseph, Salman Khan, Fahad Shahbaz Khan, and Vineeth N Balasubramanian. Towards open world object detection. In *IEEE/CVF Conference on Computer Vision and Pattern Recognition*, pages 5830–5840, 2021.

[17] Evangelos Kalogerakis, Siddhartha Chaudhuri, Daphne Koller, and Vladlen Koltun. A probabilistic model for component-based shape synthesis. *ACM Transactions on Graphics*, 31(4):1–11, 2012.

[18] Xu Keyulu, Hu Weihua, Leskovec Jure, and Jegelka Stefanie. How powerful are graph neural networks? In *International Conference on Learning Representations*. OpenReview.net, 2019.

[19] Jonathan Krause, Michael Stark, Jia Deng, and Li Fei-Fei. 3d object representations for fine-grained categorization. In *International Conference on Computer Vision Workshop*, pages 554–561, 2013.

[20] Jun Li, Chengjie Niu, and Kai Xu. Learning part generation and assembly for structure-aware shape synthesis. In *AAAI conference on artificial intelligence*, pages 11362–11369, 2020.

[21] Jun Li, Kai Xu, Siddhartha Chaudhuri, Ersin Yumer, Hao Zhang, and Leonidas Guibas. Grass: Generative recursive autoencoders for shape structures. *ACM Transactions on Graphics*, 36(4):1–14, 2017.

[22] Qi Liang, Mengmeng Xiao, and Dan Song. 3d shape recognition based on multi-modal information fusion. *Multimedia Tools and Applications*, 80:16173–16184, 2021.

[23] Minghua Liu, Ruoxi Shi, Kaiming Kuang, Yinhao Zhu, Xuanlin Li, Shizhong Han, Hong Cai, Fatih Porikli, and Hao Su. Openshape: Scaling up 3d shape representation towards open-world understanding. *Annual Conference on Neural Information Processing Systems*, 36, 2024.

[24] Katia Lupinetti, Jean-Philippe Pernot, Marina Monti, and Franca Giannini. Content-based cad assembly model retrieval: Survey and future challenges. *Computer-Aided Design*, 113:62–81, 2019.

[25] Kaichun Mo, Paul Guerrero, Li Yi, Hao Su, Peter Wonka, Niloy J Mitra, and Leonidas J Guibas. Structurenet: hierarchical graph networks for 3d shape generation. *ACM Transactions on Graphics*, 38(6):1–19, 2019.

[26] Weizhi Nie, Qi Liang, An-An Liu, Zhendong Mao, and Yangyang Li. Mmjn: Multi-modal joint networks for 3d shape recognition. In *ACM International Conference on Multimedia*, pages 908–916, 2019.

[27] Jitendra Parmar, Satyendra Chouhan, Vaskar Raychoudhury, and Santosh Rathore. Open-world machine learning: applications, challenges, and opportunities. *ACM Computing Surveys*, 55(10):1–37, 2023.

[28] Charles R Qi, Hao Su, Kaichun Mo, and Leonidas J Guibas. Pointnet: Deep learning on point sets for 3d classification and segmentation. In *IEEE/CVF Conference on Computer Vision and Pattern Recognition*, pages 652–660, 2017.

[29] Chao-Hui Shen, Hongbo Fu, Kang Chen, and Shi-Min Hu. Structure recovery by part assembly. *ACM Transactions on Graphics*, 31(6):1–11, 2012.

[30] Shaoshuai Shi, Zhe Wang, Jianping Shi, Xiaogang Wang, and Hongsheng Li. From points to parts: 3d object detection from point cloud with part-aware and part-aggregation network. *IEEE Transactions on Pattern Analysis and Machine Intelligence*, 43(8):2647–2664, 2020.

[31] Yuting Su, Yuqian Li, Weizhi Nie, Dan Song, and An-An Liu. Joint heterogeneous feature learning and distribution alignment for 2d image-based 3d object retrieval. *IEEE Transactions on Circuits and Systems for Video Technology*, 30(10):3765–3776, 2019.

[32] Sagar Vaze, Kai Han, Andrea Vedaldi, and Andrew Zisserman. Open-set recognition: A good closed-set classifier is all you need? *arXiv preprint arXiv:2110.06207*, 2021.

[33] Kai Wang, Paul Guerrero, Vladimir G Kim, Siddhartha Chaudhuri, Minhyuk Sung, and Daniel Ritchie. The shape part slot machine: Contact-based reasoning for generating 3d shapes from parts. In *European Conference on Computer Vision*, pages 610–626. Springer, 2022.

[34] Yunhai Wang, Shmulik Asafi, Oliver Van Kaick, Hao Zhang, Daniel Cohen-Or, and Baoquan Chen. Active co-analysis of a set of shapes. *ACM Transactions on Graphics*, 31(6):1–10, 2012.

[35] Xin Wei, Ruixuan Yu, and Jian Sun. View-gcn: View-based graph convolutional network for 3d shape analysis. In *IEEE/CVF Conference on Computer Vision and Pattern Recognition*, pages 1850–1859, 2020.

[36] Yiling Wu, Shuhui Wang, and Qingming Huang. Multi-modal semantic autoencoder for cross-modal retrieval. *Neurocomputing*, 331:165–175, 2019.

[37] Xianghao Xu, Paul Guerrero, Matthew Fisher, Siddhartha Chaudhuri, and Daniel Ritchie. Unsupervised 3d shape reconstruction by part retrieval and assembly. In *IEEE/CVF Conference on Computer Vision and Pattern Recognition*, pages 8559–8567, 2023.

[38] Yang Xu, Yifan Feng, and Lin Bie. Triadic elastic structure representation for open-set incremental 3d object retrieval. In *ACM International Conference on Multimedia Retrieval*, pages 20–28, 2024.

[39] Yang Xu, Yifan Feng, and Yue Gao. Negative prompt driven complementary parallel representation for open-world 3d object retrieval. In *International Joint Conference on Artificial Intelligence*, pages 1498–1506, 2024.

[40] Yang Xu, Yifan Feng, and Yu Jiang. Structure-aware residual-center representation for self-supervised open-set 3d cross-modal retrieval. In *IEEE International Conference on Multimedia and Expo*, pages 1–6, 2024.

[41] Xi Yang, Ding Xia, Taichi Kin, and Takeo Igarashi. Intra: 3d intracranial aneurysm dataset for deep learning. In *IEEE/CVF Conference on Computer Vision and Pattern Recognition*, pages 2656–2666, 2020.

[42] Li Yi, Vladimir G Kim, Duygu Ceylan, I-Chao Shen, Mengyan Yan, Hao Su, Cewu Lu, Qixing Huang, Alla Sheffer, and Leonidas Guibas. A scalable active framework for region annotation in 3d shape collections. *ACM Transactions on Graphics*, 35(6):1–12, 2016.

[43] Haoxuan You, Yifan Feng, Rongrong Ji, and Yue Gao. Pvnet: A joint convolutional network of point cloud and multi-view for 3d shape recognition. In *ACM International Conference on Multimedia*, pages 1310–1318, 2018.

[44] Haoxuan You, Yifan Feng, Xibin Zhao, Changqing Zou, Rongrong Ji, and Yue Gao. Pvrnet: Point-view relation neural network for 3d shape recognition. In *AAAI Conference on Artificial Intelligence*, pages 9119–9126, 2019.

[45] Guanqi Zhan, Qingnan Fan, Kaichun Mo, Lin Shao, Baoquan Chen, Leonidas J Guibas, Hao Dong, et al. Generative 3d part assembly via dynamic graph learning. *Annual Conference on Neural Information Processing Systems*, 33:6315–6326, 2020.

[46] Da-Wei Zhou, Han-Jia Ye, and De-Chuan Zhan. Learning placeholders for open-set recognition. In *IEEE/CVF Conference on Computer Vision and Pattern Recognition*, pages 4401–4410, 2021.

[47] Zhi-Hua Zhou. Open-environment machine learning. *National Science Review*, 9(8):nwac123, 2022.

[48] Chenming Zhu, Wenwei Zhang, Tai Wang, Xihui Liu, and Kai Chen. Object2scene: Putting objects in context for open-vocabulary 3d detection. *arXiv preprint arXiv:2309.09456*, 2023.

## A   Assembly-Based Open-Set 3DOR

In practical scenarios, the categories of 3D objects can be labeled from various perspectives such as geometric and semantic, or from different levels of granularity. From a geometric perspective, a 3D object is naturally composed of multiple part-level shapes. For instance, a 3D model of a car can be deconstructed into several part-level shapes, such as wheels, chassis, doors, and windows. Each of these parts represents a distinct geometric part that contributes to the formation of the complete 3D object. This modular decomposition facilitates more efficient manipulation, analysis, and reconstruction, as each part can be independently modified or replaced while preserving the overall structural integrity of the automobile. These parts not only enhance the hierarchical structure and diversity of labels but also play a crucial role in the representation of 3D models. Consequently, they find applications in various fields, such as computer-aided design (CAD), where precise modeling of individual components is essential; virtual reality (VR), which relies on detailed and interactive 3D environments; and

Table 4: The category splitting on seen and unseen categories for the three datasets.

| Dataset | Seen Categories (Training Set) | Unseen Categories (Testing Set) |
|---|---|---|
| OP-SHNP | airplane, guitar, lamp, laptop, mug, skateboard | bag, cap, car, chair, earphone, knife, motorbike, pistol, rocket, table |
| OP-INTRA | candelabra, goblets, telealiens | chairs, fourleg, guitars, irons, lamps, vases |
| OP-COSEG | mixed | aneurysm, vessel |

Table 5: Detailed statistic of OP-SHNP.

| Category | airplane | guitar | lamp | laptop | mug | skateboard | bag | cap |
|---|---|---|---|---|---|---|---|---|
| Samples | 2690 | 787 | 1546 | 445 | 184 | 152 | 76 | 55 |
| Parts | 4 | 3 | 4 | 2 | 2 | 3 | 2 | 2 |

| Category | car | chair | earphone | knife | motorbike | pistol | rocket | table |
|---|---|---|---|---|---|---|---|---|
| Samples | 898 | 3746 | 69 | 392 | 202 | 275 | 66 | 363 |
| Parts | 4 | 4 | 3 | 2 | 6 | 3 | 3 | 3 |

Table 6: Detailed statistic of OP-COSEG.

| Category | candelabra | goblets | telealiens | chairs | fourleg | guitar | irons | lamps | vase |
|---|---|---|---|---|---|---|---|---|---|
| Samples | 28 | 12 | 200 | 400 | 20 | 44 | 18 | 20 | 300 |
| Parts | 4 | 3 | 4 | 3 | 5 | 3 | 3 | 3 | 4 |

medical imaging, where accurate representations of anatomical structures are critical for diagnosis and treatment planning.

As for the 3D object retrieval (3DOR) task, individual parts and combinations of multiple parts can serve as crucial clues for the representation and analysis of a complete 3D shape. The greater the number of parts, the higher the accuracy and uniqueness of the object representation. This characteristic is particularly advantageous for open-set 3DOR through a *bottom-up* lens of *Part Assembly*. Given the incomplete object-level labels in open-set settings, employing a multi-layered representation and analysis of individual and associated parts can significantly enhance the accuracy of open-set 3D object retrieval (3DOR). Furthermore, it may reduce the dependency on extensive training data, addressing a common limitation in open-set environments.

## B OpenPart Dataset Generation

We generate three part-assembly driven open-set 3D object retrieval (OpenPart) datasets, including OP-SHNP, OP-INTRA, OP-COSEG based on the public dataset ShapeNetPart [42], IntrA [41], and COSEG [34]. We sampled the point cloud from the triangular surface for each dataset. Specifically, the point number for point clouds in OP-SHNP, OP-INTRA, and OP-COSEG are 2048, 2048, and 1024, respectively. In the OpenPart datasets, we use the part

Table 7: Detailed statistic of OP-INTRA.

| Category | mixed | aneurysm | vessel |
|---|---|---|---|
| Samples | 116 | 1290 | 1028 |
| Parts | 3 | 2 | 2 |

segmentation annotations of the original dataset, the classes are split into seen and unseen classes for training and testing as shown in Tab. 4. Besides, we provide more detailed statistics of the three datasets in Table 5, Table 6, and Table 7.

## C Implemental Details

The proposed Hypergraph-Based Assembly Fuzzy Representation (HAFR) framework is composed of two modules: *Isomorphic Assembly Embedding (IAE)* and *Structured Fuzzy Reconstruction (SFR)*. The basic features for input are extracted by PointNet [28] and each part feature is obtained by the average point feature of each region, we use the top four parts for each object in this paper. The IAE utilizes Hypergraph Isomorphism Convolution (HIConv) and assembly auto-encoders $\mathcal{A}_a$ to generate assembly embeddings from the basic part features. The implemented details of the IAE and SFR modules are provided in Algorithm 1 and Algorithm 2, respectively.

**Algorithm 1** Training the IAE module

---

**Input**: Basic part features $\{f^r\}_{k=1}^P$.
**Parameter**: $\alpha = 0.5$.
**Output**: Assembly embeddings $\{c_i\}_{i=1}^N$.

1: Let $epoch = 0$;
2: Initialize assembly hypergraph $\mathcal{G}_a = \{\mathbf{X}_h, \mathcal{E}_o\}$;
3: Construct vertices $\mathbf{X}_f = \bigcup_{d=1}^D \{f_i^d\}_{i=1}^N$;
4: Construct isomorphism hyperedge $\mathcal{E}_o = \{\mathcal{O}_v \mid i \in \{1, \cdots, N\}\}$;
5: Calculate diagonal degree matrices $\mathbf{D}_v$ and $\mathbf{D}_e$;
6: Calculate incidence matrix $\mathbf{H}$ of $\mathcal{G}_a$;
7: Initialize learnable HIConv parameters $\mathbf{\Theta}_{HIConv}$ of $\mathcal{G}_a$;
8: Initialize assembly auto-encoder $\mathcal{A}_a = \{\Psi, \Phi\}$;;
9: Initialize aggregation function $\mathcal{B}$;
10: **while** $epoch \leq 40$ **do**
11:     Isomorphism embeddings $\tilde{\mathbf{X}}_f = \text{MLP}((1+\epsilon)\mathbf{X}_f + \sigma(\mathbf{D}_v^{-\frac{1}{2}}\mathbf{H}\mathbf{D}_e^{-1}\mathbf{H}^\top\mathbf{D}_v^{-\frac{1}{2}}\mathbf{X}_f\mathbf{\Theta}_{HIConv}))$;
12:     Reshape isomorphism embeddings $\{\{c_i^r\}_{r=1}^P\}_{i=1}^N = \tilde{\mathbf{X}}_f$;
13:     Get unified embeddings of each part $\{\{u_i^r\}_{r=1}^P\}_{i=1}^N = \Psi^r(\{\{c_i^r\}_{r=1}^P\}_{i=1}^N)$.
14:     Get assembly embeddings of each object $\{u_i\}_{i=1}^N = \{\mathcal{B}(\{u_i^r\}_{r=1}^P)\}_{i=1}^N$
15:     Get mixed embeddings if each part $\{\{\hat{r}_i^r\}_{r=1}^P\}_{i=1}^N = \Psi^r(\{\{c_i^r\}_{r=1}^P\}_{i=1}^N)$.
16:     Calculate the Assembly Loss $\mathcal{L}_{as} = \frac{2}{R(R-1)}\sum_{k=1}^R\sum_{l=k+1}^R\|u_i^k - u_i^l\|_2$.
17:     Calculate the Cross-Part Loss $\mathcal{L}_{xp} = \frac{2}{R(R-1)}\sum_{k=1,l\neq k}^D \left(\|c_i^k - \hat{c}_i^k\|_2 + \|c_i^k - \Phi^l\left(\Psi^k\left(c_i^l\right)\right)\|_2\right)$.
18:     Calculate loss for the IAE module $\mathcal{L}_{IAE} = \alpha\mathcal{L}_{as} + (1-\alpha)\mathcal{L}_{xp}$.
19:     **if** $\mathcal{L}_{IAE}$ does not converges **then**
20:         Update parameters of $\mathbf{\Theta}_{HIConv}$ and $\mathcal{A}_a$ by $\mathcal{L}_{IAE}$.
21:         $epoch += 1$
22:     **else**
23:         Break.
24:     **end if**
25: **end while**
26: **return** Assembly embeddings $\{u_i\}_{i=1}^N$

---

Our experiments were conducted on a Tesla V100-32G GPU and an Intel(R) Xeon(R) Silver 4210 CPU @ 2.20GHz. The hyper-parameters "k" in the SFR module are set to 20, 6, and 40 for OP-SHNP, OP-INTRA, and OP-COSEG, respectively.

## D  More Results

Since no methods are specifically designed for part-assembly driven open-set 3DOR, we refine the current state-of-the-art close-set 3DOR methods (MMJN [26], TCL [13], SDML [14], MMSAE [36]), and open-set 3D learning methods (PROSER [46], HGM²R [8]), then we refine the multi-modal fusion module with multi-part fusion for each method. Specifically, these compared methods are implemented by:

**MMJN** [26]: MMJN is a multi-modal joint network that employs weighted fusion to integrate features across multiple modalities for retrieval. Specifically, we

Table 8: The hyper-parameters of the HAFR framework.

|  | IAE | SFR |
|---|---|---|
| Optimizer | SGD | SGD |
| Learning Rate | 0.1 | 0.001 |
| Momentum | 0.9 | 0.9 |
| Weight Decay | 0.0001 | 0 |
| LR Scheduler | Cosine Annealing | Cosine Annealing |
| $T_{max}$ | 40 | 60 |
| $eta_{min}$ | 0.00001 | 0.00001 |
| Max Epoches | 40 | 30 |

generate the assembly embeddings by auto-encoders and utilize them in the classification fusion part of the MMJN network.

---

**Algorithm 2** Training the SFR module

---

**Input**: Assembly embeddings $\{u_i\}_{i=1}^N$.
**Parameter**: $\beta = 0.9$.
**Output**: Fuzzy embeddings $\{z_i\}_{i=1}^N$.

 1: Let $epoch = 0$;
 2: Initialize leverage hypergraph $\mathcal{G}_{lev} = \{\mathbf{X}_u, \mathcal{E}_{lev}\}$;
 3: Construct vertices $\mathbf{X}_u = \{u_i\}_{i=1}^N$;
 4: Construct local-uncertainty hyperedge $\mathcal{E}_c = \{C_v(y) \mid y \in \mathcal{Y}\}$;
 5: Construct global-uncertainty hyperedge $\mathcal{E}_u = \{K_{\mathrm{KNN}_k}(v) \mid v \in \mathcal{V}\}$;
 6: Construct hyperedges $\mathcal{E}_{lev} = \mathcal{E}_c \cup \mathcal{E}_u$;
 7: Calculate diagonal degree matrices $\mathbf{D}_v$ and $\mathbf{D}_e$;
 8: Calculate incidence matrix $\mathbf{H}$ of $\mathcal{G}_{lev}$;
 9: Initialize HGNNConv parameters $\mathbf{\Theta}_{lev}$ of $\mathcal{G}_{lev}$;
10: Construct memory bank $\mathcal{M}$;
11: **while** $epoch \leq 120$ **do**
12:     Get propagation embeddings $\tilde{\mathbf{X}}_u = \sigma \left( \mathbf{D}_v^{-\frac{1}{2}} \mathbf{H} \mathbf{D}_e^{-1} \mathbf{H}^\top \mathbf{D}_v^{-\frac{1}{2}} \mathbf{X}_u \mathbf{\Theta}_{lev} \right)$;
13:     Reshape propagation embeddings $\{p_i\}_{i=1}^N = \tilde{\mathbf{X}}_u$;
14:     Get activation $s_{i,j} = \|\tilde{u}_i - m_j\|_2$;
15:     Get fuzzy embeddings $\{z_i\}_{i=1}^N = \{\sum_{j=1}^L s'_{i,j} m_j\}_{i=1}^N$ ;
16:     Calculate the Cross-Entropy Loss $\mathcal{L}_{ce} = -\sum_{k=1}^L \left( n_{i,k}\log(p_{i,k}) + n_{i,k}\log(q_{i,k}) \right)$;
17:     Calculate the Fuzzy Reconstruction Loss $\mathcal{L}_{fz} = \left\| \tilde{u}_i - z_i \right\|_2$;
18:     Calculate loss for the SFR module $\mathcal{L}_{SFR} = \beta \mathcal{L}_{fz} + (1 - \beta)\mathcal{L}_{ce}$.
19:     **if** $\mathcal{L}_{SFR}$ does not converges **then**
20:         Update parameters of $\mathbf{\Theta}_{lev}$ and $\mathcal{M}$ by $\mathcal{L}_{SFR}$.
21:         $epoch+=1$
22:     **else**
23:         Break.
24:     **end if**
25: **end while**
26: **return** Fuzzy embeddings $\{z_i\}_{i=1}^N$.
27: Retrieval by fuzzy embeddings $\{z_i\}_{i=1}^N$.

---

**TCL** [13]: TCL is a method based on metric learning, combining triplet and center loss to get unified fusion embeddings from different modalities. We use the center-based method for object center embedding from different parts

**SDML** [14]: SDML is a metric-learning-based method for cross-modal retrieval, which learns projection functions for different modalities independently. We construct the assembly embeddings by auto-encoders and use the joint supervision aligned with its triplet center loss.

**MMSAE** [36]: MMSAE is a multi-modal retrieval method using auto-encoders. It trains encoders with a reconstruction loss function to align embeddings from various modalities into a unified latent space. We add auto-encoders to generate assembly embeddings, and we use them for alignment with semantic code vectors for 3D object retrieval.

**PROSER** [46]: PROSER is an open-world recognition method that extends the closed-set classifier to determine if a sample belongs to seen categories or not. We construct auto-encoders for assembly embedding, and we mix up the assembly embeddings with the multiple dummy classifiers in the PROSER.

**HGM²R** [8]: HGM²R is an open-world 3D multi-modal retrieval method, which retrieves the objects from unseen categories through structure-aware learning. We construct auto-encoders for assembly embedding of 3D parts, and we use the KNN-based hyperedges in the SAIKL module.

We provide the visualized results of assembly embeddings and fuzzy embeddings in Fig. 6 and Fig. 7. The visualizations indicate that the SFR module is capable of accurately representing open-set categories, which can effectively enhance retrieval performance.

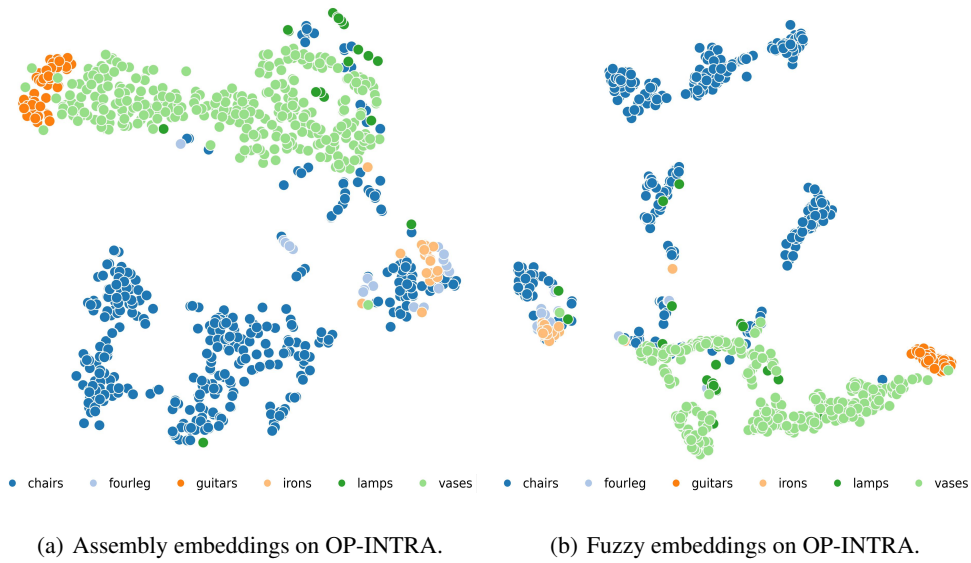

(a) Assembly embeddings on OP-INTRA.    (b) Fuzzy embeddings on OP-INTRA.

Figure 6: The t-SNE visualization of the embeddings from unseen categories in the OP-INTRA and OP-SHNP datasets.

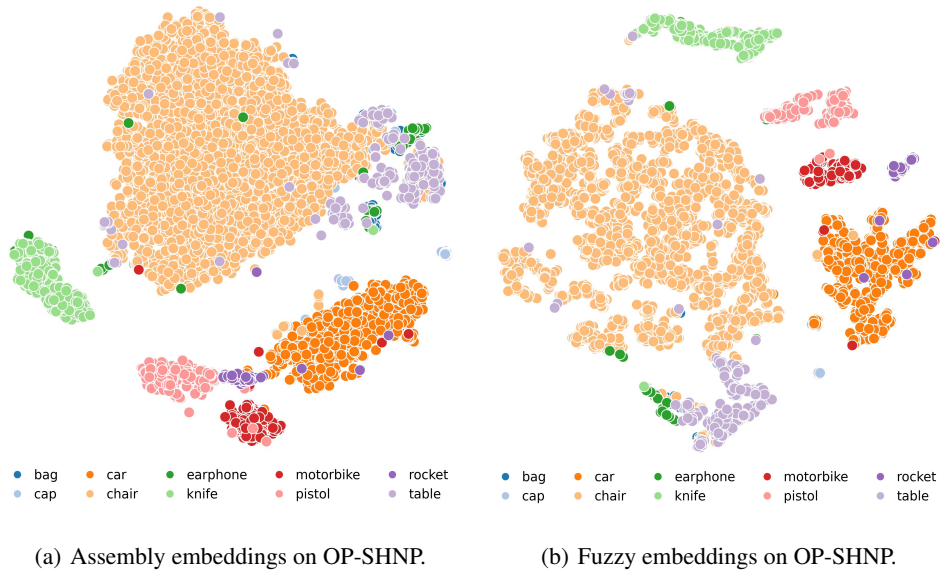

(a) Assembly embeddings on OP-SHNP.    (b) Fuzzy embeddings on OP-SHNP.

Figure 7: The t-SNE visualization of the embeddings from unseen categories in the OP-SHNP dataset.
